# Adaptive Scaling for Feature Selection in SVMs

**Yves Grandvalet**
Heudiasyc, UMR CNRS 6599,
Université de Technologie de Compiègne,
Compiègne, France
Yves.Grandvalet@utc.fr

**Stéphane Canu**
PSI
INSA de Rouen,
St Etienne du Rouvray, France
Stephane.Canu@insa-rouen.fr

## Abstract

This paper introduces an algorithm for the automatic relevance determination of input variables in kernelized Support Vector Machines. Relevance is measured by scale factors defining the input space metric, and feature selection is performed by assigning zero weights to irrelevant variables. The metric is automatically tuned by the minimization of the standard SVM empirical risk, where scale factors are added to the usual set of parameters defining the classifier. Feature selection is achieved by constraints encouraging the sparsity of scale factors. The resulting algorithm compares favorably to state-of-the-art feature selection procedures and demonstrates its effectiveness on a demanding facial expression recognition problem.

## 1 Introduction

In pattern recognition, the problem of selecting relevant variables is difficult. Optimal subset selection is attractive as it yields simple and interpretable models, but it is a combinatorial and acknowledged unstable procedure [2]. In some problems, it may be better to resort to stable procedures penalizing irrelevant variables. This paper introduces such a procedure applied to Support Vector Machines (SVM).

The relevance of input features may be measured by continuous weights or scale factors, which define a diagonal metric in input space. Feature selection consists then in determining a sparse diagonal metric, and sparsity can be encouraged by constraining an appropriate norm on scale factors. Our approach can be summarized by the setting of a global optimization problem pertaining to 1) the parameters of the SVM classifier, and 2) the parameters of the feature space mapping defining the metric in input space. As in standard SVMs, only two tunable hyper-parameters are to be set: the penalization of training errors, and the magnitude of kernel bandwiths. In this formalism we derive an efficient algorithm to monitor slack variables when optimizing the metric. The resulting algorithm is fast and stable.

After presenting previous approaches to hard and soft feature selection procedures in the context of SVMs, we present our algorithm. This exposure is followed by an experimental section illustrating its performances and conclusive remarks.

## 2 Feature Selection via adaptive scaling

Scaling is a usual preprocessing step, which has important outcomes in many classification methods including SVM classifiers [9, 3]. It is defined by a linear transformation within the input space: $\widetilde{\mathbf{x}} = \mathbf{\Sigma}\mathbf{x}$, where $\mathbf{\Sigma} = \text{diag}(\boldsymbol{\sigma})$ is a diagonal matrix $\mathbf{\Sigma}_{kk'} = \sigma_k \delta_{kk'}$ of scale factors.

Adaptive scaling consists in letting $\boldsymbol{\sigma}$ to be adapted during the estimation process with the explicit aim of achieving a better recognition rate. For kernel classifiers, $\boldsymbol{\sigma}$ is a set of hyperparameters of the learning process. According to the structural risk minimization principle [8], $\boldsymbol{\sigma}$ can be tuned in two ways:

1. estimate the parameters of classifier $f$ by empirical risk minimization for several values of $\{\sigma_k\}_{k=1}^d$ to produce a structure of classifiers $f_{\boldsymbol{\sigma}}$ multi-indexed by $\{\sigma_k\}_{k=1}^d$. Select one element of the structure by finding the set $\{\sigma_k\}_{k=1}^d$ minimizing some estimate of generalization error.

2. estimate the parameters of classifier $f$ and the hyper-parameters $\{\sigma_k\}_{k=1}^d$ by empirical risk minimization, while a second level hyper-parameter, say $\sigma_0$, constrains $\{\sigma_k\}_{k=1}^d$ in order to avoid overfitting. This procedure produces a structure of classifiers indexed by $\sigma_0$, whose value is computed by minimizing some estimate of generalization error.

The usual paradigm consists in computing the estimate of generalization error for regularly spaced hyper-parameter values and picking the best solution among all trials. Hence, the first approach requires intensive computation, since the trials should be completed over a $d$-dimensional grid over $\sigma_k$ values.

Several authors suggested to address this problem by optimizing an estimate of generalization error with respect to the hyper-parameters. For SVM classifiers, Cristianini *et al.* [4] first proposed to apply an iterative optimization scheme to estimate a single kernel width hyper-parameter. Weston *et al.* [9] and Chapelle *et al.* [3] generalized this approach to multiple hyper-parameters in order to perform adaptive scaling and variable selection.

The experimental results in [9, 3] show the benefits of this optimization. However, relying on the optimization of generalization error estimates over many hyper-parameters is hazardous. Once optimized, the unbiased estimates become down-biased, and the bounds provided by VC-theory usually hold for kernels defined *a priori* (see the proviso on the radius/margin bound in [8]). Optimizing these criteria may thus result in overfitting.

In the second solution considered here, the estimate of generalization error is minimized with respect to $\sigma_0$, a single (second level) hyper-parameter, which constrains $\{\sigma_k\}_{k=1}^d$. The role of this constraint is twofold: control the complexity of the classifier, and encourage variable selection in input space. This approach is related to some successful soft-selection procedures, such as lasso and bridge [5] in the frequentist framework and Automatic Relevance Determination (ARD) [7] in the Bayesian framework. Note that this type of optimization procedure has been proposed for linear SVM in both frequentist [1] and Bayesian frameworks [6]. Our method generalizes this approach to nonlinear SVM.

## 3 Algorithm

### 3.1 Support Vector Machines

The decision function provided by SVM is $\text{sign}(f_{\boldsymbol{\sigma}}(\mathbf{x}))$, where function $f_{\boldsymbol{\sigma}}$ is defined as:

$$f_{\boldsymbol{\sigma}}(\mathbf{x}) = \mathbf{w}^\top \boldsymbol{\phi}_{\boldsymbol{\sigma}}(\mathbf{x}) + b = \sum_i y_i \alpha_i K_{\boldsymbol{\sigma}}(\mathbf{x}_i, \mathbf{x}) + b \ , \tag{1}$$

where the parameters $(\mathbf{w}, b)$ are obtained by solving the following optimization problem:

$$\left\{ \begin{array}{ll} \displaystyle\min_{\mathbf{w}, b, \boldsymbol{\xi}} & \dfrac{1}{2}\mathbf{w}^{\top}\mathbf{w} + C\sum_{i=1}^{n}\xi_i \\ \text{subject to} & y_i(\mathbf{w}^{\top}\boldsymbol{\phi}_{\boldsymbol{\sigma}}(\mathbf{x}_i) + b) \geq 1 - \xi_i \quad i = 1, \ldots, n \\ & \xi_i \geq 0 \qquad\qquad\qquad\qquad\qquad i = 1, \ldots, n \ . \end{array} \right. \tag{2}$$

with $\boldsymbol{\phi}_{\boldsymbol{\sigma}}(\mathbf{x})$ defined as $\phi(\boldsymbol{\Sigma}\mathbf{x})$. In this problem setting, $C$ and the parameters $\boldsymbol{\sigma}$ of the feature space mapping (typically a kernel bandwidth) are tunable hyper-parameters which need to be determined by the user.

## 3.2 A global optimization problem

In [9, 3], adaptive scaling is performed by iteratively finding the parameters $(\mathbf{w}, b)$ of the SVM classifier $f_{\boldsymbol{\sigma}}$ for a fixed value of $\boldsymbol{\sigma} = \{\sigma_k\}_{k=1}^{d}$ and minimizing a bound on the estimate of generalization error with respect to hyper-parameters $(\{\sigma_k\}_{k=1}^{d}, C)$. The algorithm minimizes 1) the SVM empirical criterion with respect to parameters and 2) an estimate of generalization error with respect to hyper-parameters.

In the present approach, we avoid the enlargement of the set of hyper-parameters by letting $\{\sigma_k\}_{k=1}^{d}$ to be standard parameters of the classifier. Complexity is controlled by $C$ and by constraining the magnitude of $\boldsymbol{\sigma}$. The latter defines the single hyper-parameter of the learning process related to scaling variables. The learning criterion is defined as follows:

$$\left\{ \begin{array}{ll} \displaystyle\min_{\boldsymbol{\sigma}}\min_{\mathbf{w}, b, \boldsymbol{\xi}} & \dfrac{1}{2}\mathbf{w}^{\top}\mathbf{w} + C\sum_{i=1}^{n}\xi_i \\ \text{subject to} & y_i(\mathbf{w}^{\top}\boldsymbol{\phi}_{\boldsymbol{\sigma}}(\mathbf{x}_i) + b) \geq 1 - \xi_i \quad i = 1, \ldots, n \\ & \xi_i \geq 0 \qquad\qquad\qquad\qquad\qquad i = 1, \ldots, n \\ & \dfrac{1}{d}\sum_{k=1}^{d}\sigma_k^p = \sigma_0^p \\ & \sigma_k \geq 0 \qquad\qquad\qquad\qquad\quad k = 1, \ldots, d \end{array} \right. \tag{3}$$

Like in standard SVM classification, the minimization of an estimate of generalization error is postponed to a later step, which consists in picking the best solution among all trials on the two dimensional grid of hyper-parameters $(\sigma_0, C)$.

In (3), the constraint on $\boldsymbol{\sigma}$ should favor sparse solutions. To allow $\sigma_k$ to go to zero, $p$ should be positive. To encourage sparsity, zeroing a small $\sigma_k$ should allow a high increase of $\sigma_l$, $l \neq k$, hence $p$ should be small. In the limit of $p \to 0$, the constraint counts the number of non-zero scale parameters, resulting in a hard selection procedure. This choice might seem appropriate for our purpose, but it amounts to attempt to solve a highly non-convex optimization problem, where the number of local minima grows exponentially with the input dimension $d$. To avoid this problem, we suggest to use $p = 2$, which is the smallest value for which the problem is convex with the linear mapping $\boldsymbol{\phi}_{\boldsymbol{\sigma}}(\mathbf{x}) = \boldsymbol{\Sigma}\mathbf{x}$. Indeed, for linear kernels, the constraint on $\boldsymbol{\sigma}$ amounts to minimize the standard SVM criterion where the penalization on the $\ell^2$ norm is replaced by the penalization of the $\ell^{\frac{2p}{p+2}}$ norm. Hence, setting $p = 2$ provides the solution of the $\ell^1$ SVM classifier described in [1]. For non-linear kernels however, the two solutions differ notably since the present algorithm modifies the metric in input space, while the $\ell^1$ SVM classifier modifies the metric in feature space. Finally, note that unicity can be guaranteed for $p = 2$ and Gaussian kernels with large bandwidths ($\sigma_0 \to 0$).

### 3.3 An alternated optimization scheme

Problem (3) is complex; we propose to solve iteratively a series of simplier problems. The function $f$ is first optimized with respect to parameters $(\mathbf{w}, b)$ for a fixed mapping $\phi_{\boldsymbol{\sigma}}$ (standard SVM problem). Then, the parameters $\boldsymbol{\sigma}$ of the feature space mapping are optimized while some characteristics of $f$ are kept fixed: At step $s$, starting from a given $\boldsymbol{\sigma}^{(s)}$ value, the optimal $(\widehat{\mathbf{w}}(\boldsymbol{\sigma}^{(s)}), \widehat{b}(\boldsymbol{\sigma}^{(s)}))$ are computed. Then $\boldsymbol{\sigma}^{(s+1)}$ is determined by a descent algorithm.

In this scheme, $(\widehat{\mathbf{w}}(\boldsymbol{\sigma}^{(s)}), \widehat{b}(\boldsymbol{\sigma}^{(s)}))$ are computed by solving the standard quadratic optimization problem (2). Our implementation, based on an interior point method, will not be detailed here. Several SVM retraining are necessary, but they are faster than the usual training since the algorithm is initialized appropriately with the solutions of the preceding round.

For solving the minimization problem with respect to $\boldsymbol{\sigma}$, we use a reduced conjugate gradient technique. The optimization problem was simplified by assuming that some of the other variables are fixed. We tried several versions: 1) $\mathbf{w}$ fixed; 2) Lagrange multipliers $\boldsymbol{\alpha}$ fixed; 3) set of support vectors fixed. For the three versions, the optimal value of $b$, or at least the optimal value of the slack variables $\boldsymbol{\xi}$ can be obtained by solving a linear program, whose optimum is computed directly (in a single iteration). We do not detail our first version here, since the two last ones performed much better. The main steps of the two last versions are sketched below.

### 3.4 Sclaling parameters update

Starting from an initial solution $(\boldsymbol{\sigma}, \mathbf{w}(\boldsymbol{\sigma}), b(\boldsymbol{\sigma}))$, our goal is to update $\boldsymbol{\sigma}$ by solving a simple intermediate problem providing an improved solution to the global problem (3). We first assume that the Lagrange multipliers $\boldsymbol{\alpha}$ defining $\mathbf{w}$ are not affected by $\boldsymbol{\sigma}$ updates, so that $\mathbf{w}$ is defined as $\mathbf{w} = \sum_{j=1}^{n} \alpha_j y_j \phi_{\boldsymbol{\sigma}}(\mathbf{x}_i)$.

Regarding problem (3), $\mathbf{w}$ is sub-optimal when $\boldsymbol{\sigma}$ varies; nevertheless $\mathbf{w}$ is guaranteed to be an admissible solution. Hence, we minimize an upper bound of the original primal cost which guarantees that any admissible update (providing a decrease of the cost) of the intermediate problem will provide a decrease of the cost of the original problem.

The intermediate optimization problem is stated as follows:

$$
\left\{
\begin{array}{lll}
\displaystyle \min_{\boldsymbol{\sigma}, b, \boldsymbol{\xi}} & \displaystyle \frac{1}{2} \sum_{i,j} \alpha_i \alpha_j y_i y_j K_{\boldsymbol{\sigma}}(\mathbf{x}_i, \mathbf{x}_j) + C \sum_{i=1}^{n} \xi_i \\[4mm]
\text{subject to} & \displaystyle y_i \left( \sum_{i,j} \alpha_j y_j K_{\boldsymbol{\sigma}}(\mathbf{x}_i, \mathbf{x}_j) + b \right) \geq 1 - \xi_i & i = 1, \ldots, n \\[4mm]
& \xi_i \geq 0 & i = 1, \ldots, n \\[4mm]
& \displaystyle \frac{1}{d} \sum_{k=1}^{d} \sigma_k^p = \sigma_0^p \\[4mm]
& \sigma_k \geq 0 & k = 1, \ldots, d \ .
\end{array}
\right.
\tag{4}
$$

Solving this problem is still difficult since the cost is a complex non-linear function of scale factors. Hence, as stated above, $\boldsymbol{\sigma}$ will be updated by a descent algorithm. The latter requires the evaluation of the cost and its gradient with respect to $\boldsymbol{\sigma}$. In particular, this means that we should be able to compute $\sum_{i=1}^{n} \xi_i$ and $\partial \sum_{i=1}^{n} \xi_i / \partial \boldsymbol{\sigma}$ for any value of $\boldsymbol{\sigma}$.

For given values of $\boldsymbol{\sigma}$ and $\boldsymbol{\alpha}$, $\boldsymbol{\xi}$ is the solution of the following problem:

$$\begin{cases} \displaystyle\min_{b,\boldsymbol{\xi}} \quad C\sum_{i=1}^{n}\xi_i \\[2ex] \text{subject to} \quad y_i\left(\sum_{j=1}^{n}\alpha_j y_j K_{\boldsymbol{\sigma}}(\mathbf{x}_i,\mathbf{x}_j)+b\right) \geq 1-\xi_i \quad i=1,\ldots,n \\[2ex] \qquad\qquad \xi_i \geq 0 \qquad\qquad\qquad\qquad\qquad\qquad\qquad i=1,\ldots,n \;, \end{cases} \tag{5}$$

whose dual formulation is

$$\begin{cases} \displaystyle\max_{\boldsymbol{\mu}} \quad \sum_{i=1}^{n}\mu_i\left\{1-y_i\left(\sum_{j=1}^{n}\alpha_j y_j K_{\boldsymbol{\sigma}}(\mathbf{x}_i,\mathbf{x}_j)\right)\right\} \\[2ex] \text{subject to} \quad \sum_{i=1}^{n}\mu_i y_i = 0 \\[1ex] \qquad\qquad C \geq \mu_i \geq 0 \qquad\qquad\qquad\qquad\qquad\qquad i=1,\ldots,n \;. \end{cases} \tag{6}$$

This linear problem is solved directly by the following algorithm: 1) sort $1-y_i\left(\sum_{j=1}^{n}\alpha_j y_j K(\mathbf{x}_i,\mathbf{x}_j)\right)$ in descending order for all positive examples on the one side and for all negative examples on the other side; 2) compute the pairwise sum of sorted values; 3) set $\mu_i = C$ for all positive and negative examples whose sum is positive.

With $\boldsymbol{\mu}$, $C\sum_{i=1}^{n}\xi_i$ and its derivative with respect to $\boldsymbol{\sigma}$ are easily computed. Parameters $\boldsymbol{\sigma}$ are then updated by a conjugate reduced gradient technique, *i.e.* a conjugate gradient algorithm ensuring that the set of constraints on $\boldsymbol{\sigma}$ are always verified.

### 3.5 Updating Lagrange multipliers

Assume now that only the support vectors remain fixed while optimizing $\boldsymbol{\sigma}$. This assumption is used to derive a rule to update at reasonable computing cost the Lagrange multipliers $\boldsymbol{\alpha}$ together with $\boldsymbol{\sigma}$ by computing $\partial\boldsymbol{\alpha}/\partial\boldsymbol{\sigma}$. At $(\boldsymbol{\alpha},\boldsymbol{\sigma},b)$, the following holds [3]:

1. for support vectors of the first category $f_{\boldsymbol{\sigma}}(\mathbf{x}_i)=y_i$

$$\sum_{j=1}^{n}\alpha_j y_j K_{\boldsymbol{\sigma}}(\mathbf{x}_i,\mathbf{x}_j)+b = y_i \;; \tag{7}$$

2. for support vectors of the second category (such that $\xi_i > 0$) $\alpha_i = C$.

From these equations, and the assumption that support vectors remain support vectors (and that their category do not change) one derives a system of linear equations defining the derivatives of $\boldsymbol{\alpha}$ and $b$ with respect to $\boldsymbol{\sigma}$ [3]:

1. for support vectors of the first category

$$\sum_{j=1}^{n}\frac{\partial\alpha_j}{\partial\boldsymbol{\sigma}}y_j K_{\boldsymbol{\sigma}}(\mathbf{x}_i,\mathbf{x}_j)+\sum_{j=1}^{n}\alpha_j y_j \nabla_{\boldsymbol{\sigma}} K_{\boldsymbol{\sigma}}(\mathbf{x}_i,\mathbf{x}_j)+\frac{\partial b}{\partial\boldsymbol{\sigma}}=0 \tag{8}$$

2. for support vectors of the second category $\dfrac{\partial\alpha_i}{\partial\boldsymbol{\sigma}}=0$

3. Finally, the system is completed by stating that the Lagrange multipliers should obey the constraint $\sum_{j=1}^{n} \alpha_j y_j = 0$:

$$\sum_{j=1}^{n} \frac{\partial \alpha_j}{\partial \boldsymbol{\sigma}} y_j = 0 \qquad (9)$$

The value of $\boldsymbol{\alpha}$ is updated from these equations, and the step size is limited to ensure that $C > \alpha_i > 0$ for support vectors of the first category. Hence, in this version, $\mathbf{w}$ is also an admissible sub-optimal solution regarding problem (3).

## 4 Experiments

In the experiments reported below, we used $p = 2$ for the constraint on $\boldsymbol{\sigma}$ (3). The scale parameters were optimized with the last version, where the set of support vectors is assumed to be fixed. Finally, the hyper-parameters $(\sigma_0, C)$ were chosen using the span bound [3]. Although the value of the bound itself was not a faithful estimate of test error, the average loss induced by using the minimizer of these bounds was quite small.

### 4.1 Toy experiment

In [9], Weston *et al.* compared two versions of their feature selection algorithm, to standard SVMs and filter methods (*i.e.* preprocessing methods selecting features either based on Pearson correlation coefficients, Fisher criterion score, or the Kolmogorov-Smirnov statistic). Their artificial data benchmarks provide a basis for comparing our approach with their, which is based on the minimization of error bounds. Two types of distributions are provided, whose detailed characteristics are not given here. In the linear problem, 6 dimensions out of 202 are relevant. In the nonlinear problem, two features out of 52 are relevant. For each distribution, 30 experiments are conducted, and the average test recognition rate measures the performance of each method.

For both problems, standard SVM achieve a 50% error rate in the considered range of training set sizes. Our results are shown in Figure 1.

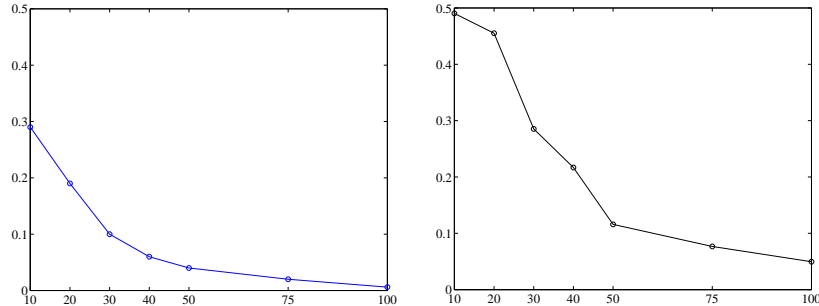

Figure 1: Results obtained on the benchmarks of [9]. Left: linear problem; right nonlinear problem. The number of training examples is represented on the $x$-axis, and the average test error rate on the $y$-axis.

Our test performances are qualitatively similar to the ones obtained by gradient descent on the radius/margin bound in [9], which are only improved by the forward selection algorithm

minimizing the span bound. Note however that Weston *et al.* results are obtained after a correct number of features was specified by the user, whereas the present results were obtained fully automatically. Knowing the number of features that should be selected by the algorithm is somewhat similar to select the optimal value of parameter $p$ for each $\sigma_0$.

In the non-linear problem, for $n = 10$ training examples, an average of 26.5 features are selected; for $n = 100$, an average of 6.6 features are selected. These figures show that although our feature selection scheme is effective, it should be more stringent: a smaller value of $p$ would be more appropriate for this type of problem. The two relevant variables are selected in 47% of cases for $n = 10$, in 90% for n=50, and in 100% for $n = 75$ and $n = 100$. For these two sample sizes, they are even always ranked first and second.

Regarding training times, the optimization of $\sigma$ required an average of over 100 times more computing time than standard SVM fitting for the linear problem and 40 times for the nonlinear problem. These increases scale less than linearly with the number of variables, and are certainly yet to be improved.

## 4.2 Expression recognition

We also tested our algorithm on a more demanding task to test its ability to handle a large number of features. The considered problem consists in recognizing the happiness expression among the five other facial expressions corresponding to universal emotions (disgust, sadness, fear, anger, and surprise). The data sets are made of $70 \times 60$ gray level images of frontal faces, with standardized positions of eyes, nose and mouth. The training set comprises 60 positive images, and 180 negative ones. The test set is made of 40 positive images and 110 negative ones.

We used the raw pixel representation of images, resulting in 4200 highly correlated features. For this task, the accuracy of standard SVMs is 92.6% (11 test errors). The recognition rate is not significantly affected by our feature selection scheme (10 errors), but more than 1300 pixels are considered to be completely irrelevant at the end of the iterative procedure (estimating $\sigma$ required about 80 times more computing time than standard SVM). This selection brings some important clues for building relevant attributes for the facial recognition expression task.

Figure 2 represents the scaling factors $\sigma$, where black is zero and white represents the highest value. We see that, according to the classifier, the relevant areas for recognizing the happiness expression are mainly in the mouth area, especially on the mouth wrinkles, and to a lesser extent in the white of the eyes (which detects open eyes) and the outer eyebrows. On the right hand side of this figure, we displayed masked support faces, *i.e.* support faces scaled by the expression mask. Although we lost many important features regarding the identity of people, the expression is still visible on these faces. Areas irrelevant for the recognition task (forehead, nose, and upper cheeks) have been erased or softened by the expression mask.

## 5 Conclusion

We have introduced a method to perform automatic relevance determination and feature selection in nonlinear SVMs. Our approach considers that the metric in input space defines a set of parameters of the SVM classifier. The update of the scale factors is performed by iteratively minimizing an approximation of the SVM cost. The latter is efficiently minimized with respect to slack variables when the metric varies. The approximation of the cost function is tight enough to allow large update of the metric when necessary. Furthermore, because at each step our algorithm guaranties the global cost to decrease, it is stable.

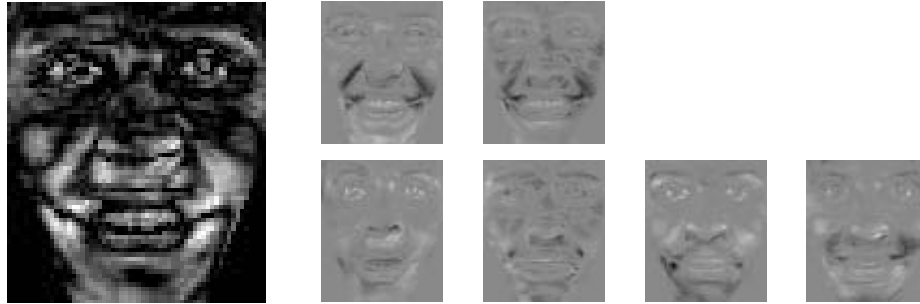

Figure 2: Left: expression mask of happiness provided by the scaling factors $\sigma$; Right, top row: the two positive masked support face; Right, bottom row: four negative masked support faces.

Preliminary experimental results show that the method provides sensible results in a reasonable time, even in very high dimensional spaces, as illustrated on a facial expression recognition task. In terms of test recognition rates, our method is comparable with [9, 3]. Further comparisons are still needed to demonstrate the practical merits of each paradigm.

Finally, it may also be beneficial to mix the two approaches: the method of Cristianini *et al.* [4] could be used to determine $\sigma_0$ and $C$. The resulting algorithm would differ from [9, 3], since the relative relevance of each feature (as measured by $\sigma_k/\sigma_0$) would be estimated by empirical risk minimization, instead of being driven by an estimate of generalization error.

## References

[1] P. S. Bradley and O. L. Mangasarian. Feature selection via concave minimization and support vector machines. In *Proc. 15th International Conf. on Machine Learning*, pages 82–90. Morgan Kaufmann, San Francisco, CA, 1998.

[2] L. Breiman. Heuristics of instability and stabilization in model selection. *The Annals of Statistics*, 24(6):2350–2383, 1996.

[3] O. Chapelle, V. Vapnik, O. Bousquet, and S. Mukherjee. Choosing multiple parameters for support vector machines. *Machine Learning*, 46(1):131–159, 2002.

[4] N. Cristianini, C. Campbell, and J. Shawe-Taylor. Dynamically adapting kernels in support vector machines. In M. S. Kearns, S. A. Solla, and D. A. Cohn, editors, *Advances in Neural Information Processing Systems 11*. MIT Press, 1999.

[5] T. Hastie, R. Tibshirani, and J. Friedman. *The Elements of Statistical Learning: data mining , inference, and prediction*. Springer series in statistics. Springer, 2001.

[6] T. Jebara and T. Jaakkola. Feature selection and dualities in maximum entropy discrimination. In *Uncertainity In Artificial Intellegence*, 2000.

[7] R. M. Neal. *Bayesian Learning for Neural Networks*, volume 118 of *Lecture Notes in Statistics*. Springer, 1996.

[8] V. N. Vapnik. *The Nature of Statistical Learning Theory*. Springer Series in Statistics. Springer, 1995.

[9] J. Weston, S. Mukherjee, O. Chapelle, M. Pontil, T. Poggio, and V. Vapnik. Feature selection for SVMs. In *Advances in Neural Information Processing Systems 13*. MIT Press, 2000.
